# Causal Categorization with Bayes Nets

**Bob Rehder**
Department of Psychology
New York University
New York, NY 10012
*bob.rehder@nyu.edu*

## Abstract

A theory of categorization is presented in which knowledge of causal relationships between category features is represented as a Bayesian network. Referred to as *causal-model theory*, this theory predicts that objects are classified as category members to the extent they are likely to have been produced by a categorys causal model. On this view, people have models of the world that lead them to expect a certain distribution of features in category members (e.g., correlations between feature pairs that are directly connected by causal relationships), and consider exemplars good category members when they manifest those expectations. These expectations include sensitivity to higher-order feature interactions that emerge from the asymmetries inherent in causal relationships.

Research on the topic of categorization has traditionally focused on the problem of learning new categories given observations of category members. In contrast, the *theory-based view* of categories emphasizes the influence of the prior theoretical knowledge that learners often contribute to their representations of categories [1]. However, in contrast to models accounting for the effects of empirical observations, there have been few models developed to account for the effects of prior knowledge. The purpose of this article is to present a model of categorization referred to as *causal-model theory* or *CMT* [2, 3]. According to CMT, people's knowledge of many categories includes not only features, but also an explicit representation of the causal mechanisms that people believe link the features of many categories.

In this article I apply CMT to the problem of establishing objects category membership. In the psychological literature one standard view of categorization is that objects are placed in a category to the extent they have features that have often been observed in members of that category. For example, an object that has most of the features of birds (e.g., wings, fly, build nests in trees, etc.) and few features of other categories is thought to be a bird. This view of categorization is formalized by *prototype models* in which classification is a function of the similarity (i.e., number of shared features) between a mental representation of a category prototype and a to-be-classified object. However, a well-known difficulty with prototype models is that a features contribution to category membership is independent of the presence or absence of other features. In contrast, consideration of a categorys theoretical knowledge is likely to influence which *combinations* of features make for acceptable category members. For example, people believe that birds have nests in trees *because* they can fly, and in light of this knowledge an animal that doesnt fly

and yet still builds nests in trees might be considered a less plausible bird than an animal that builds nests on the ground and doesnt fly (e.g., an ostrich) even though the latter animal has fewer features typical of birds.

To assess whether knowledge in fact influences which feature combinations make for good category members, in the following experiment undergraduates were taught novel categories whose four binary features exhibited either a *common-cause* or a *common-effect schema* (Figure 1). In the common-cause schema, one category feature ($F_1$) is described as causing the three other features ($F_2$, $F_3$, and $F_4$). In the common-effect schema one feature ($F_4$) is described as being caused by the three others ($F_1$, $F_2$, and $F_3$). CMT assumes that people represent causal knowledge such as that in Figure 1 as a kind of Bayesian network [4] in which nodes are variables representing binary category features and directed edges are causal relationships representing the presence of *probabilistic causal mechanisms* between features. Specifically, CMT assumes that when a cause feature is present it enables the operation of a causal mechanism that will, with some probability $m$, bring about the presence of the effect feature. CMT also allow for the possibility that effect features have potential *background causes* that are not explicitly represented in the network, as represented by parameter $b$ which is the probability that an effect will be present even when its network causes are absent. Finally, each cause node has a parameter $c$ that represents the probability that a cause feature will be present.

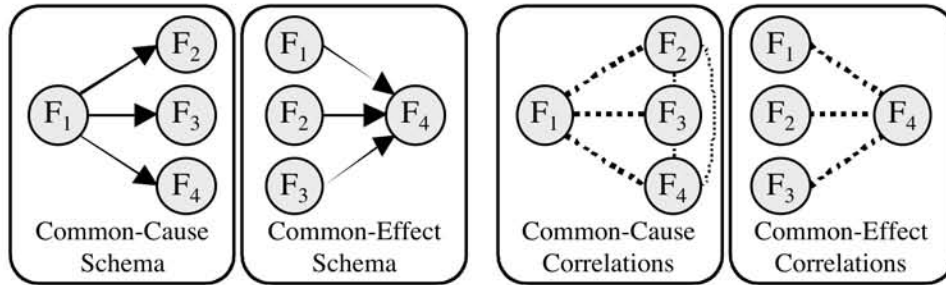

| Common-Cause Schema | Common-Effect Schema | Common-Cause Correlations | Common-Effect Correlations |

Figure 1.                                      Figure 2.

The central prediction of CMT is that an object is considered to be a category member to the extent that its features were likely to have been *generated* by a category's causal mechanisms. For example, Table 1 presents the likelihoods that the causal models of Figure 1 will generate the sixteen possible combinations of $F_1$, $F_2$, $F_3$, and $F_4$. Each likelihood equation can be derived by the application of simple Boolean algebra operations. For example, the probability of exemplar 1101 ($F_1$, $F_2$, $F_4$ present, $F_3$ absent) being generated by a common-cause model is the probability that $F_1$ is present $[c]$, times the probability that $F_2$ was brought about by $F_1$ or its background cause $[1-(1-m)(1-b)]$, times the probability that $F_3$ was brought about by neither $F_1$ nor its background cause $[(1-m)(1-b)]$, times the probability that $F_4$ was brought about by $F_1$ or its background cause $[1-(1-m)(1-b)]$. Likewise, the probability of exemplar 1011 ($F_1$, $F_3$, $F_4$ present, $F_2$ absent) being generated by a common-effect model is the probability that $F_1$ and $F_3$ are present $[c^2]$, times the probability that $F_2$ is absent $[1-c]$, times the probability that $F_4$ was brought about by $F_1$, $F_3$, or its background cause $[1-(1-m)(1-m)(1-b)]$. Note that these likelihoods assume that the causal mechanisms in each model operate independently and with the same probability $m$, restrictions that can be relaxed in other applications.

This formalization of categorization offered by CMT implies that peoples theoretical knowledge leads them to expect a certain distribution of features in category members, and that they use this information when assigning category membership. Thus, to gain insight into the categorization performance predicted by CMT, we can examine the statistical properties of category features that one can

expect to be generated by a causal model. For example, dotted lines in Figure 2 represent the features correlations that are generated from the causal schemas of Figure 1. As one would expect, pairs of features directly linked by causal relationships are correlated in the common-cause schema $F_1$ is correlated with its effects and in the common-effect schema $F_4$ is correlated with its causes. Thus, CMT predicts that combinations of features serve as evidence for category membership to the extent that they *preserve* these expected correlations (i.e., both cause and effect present or both absent), and against category membership to the extent that they *break* those correlations (one present and the other absent).

Table 1: Likelihoods Equations and Observed and Predicted Values

| Exemplar | Common Cause Schema | | | Common Effect Schema | | | Control |
|---|---|---|---|---|---|---|---|
| | Likelihood | Observed | Predicted | Likelihood | Observed | Predicted | Observed |
| 0000 | $c'b'^3$ | 60.0 | 61.7 | $c'^3 b'$ | 70.0 | 69.3 | 70.7 |
| 0001 | $c'b'^2 b$ | 44.9 | 45.7 | $c'^3 b$ | 26.3 | 27.8 | 67.0 |
| 0010 | $c'b'^2 b$ | 46.1 | 45.7 | $cc'^2 m'b'$ | 43.4 | 47.7 | 65.6 |
| 0100 | $c'b'^2 b$ | 42.8 | 45.7 | $cc'^2 m'b'$ | 47.3 | 47.7 | 66.0 |
| 1000 | $cm'^3 b'^3$ | 44.5 | 44.1 | $cc'^2 m'b'$ | 48.0 | 47.7 | 67.0 |
| 0011 | $c'b'b^2$ | 41.0 | 40.1 | $cc'^2(1-m'b')$ | 56.3 | 56.5 | 67.1 |
| 0101 | $c'b'b^2$ | 40.8 | 40.1 | $cc'^2(1-m'b')$ | 56.5 | 56.5 | 66.5 |
| 0110 | $c'b'b^2$ | 42.7 | 40.1 | $c^2 c'm'^2 b'$ | 38.3 | 39.2 | 65.6 |
| 1001 | $cm'^2 b'^2(1-m'b')$ | 55.1 | 52.7 | $cc'^2(1-m'b')$ | 57.7 | 56.5 | 68.0 |
| 1010 | $cm'^2 b'^2(1-m'b')$ | 52.6 | 52.7 | $c^2 c'm'^2 b'$ | 43.0 | 39.2 | 67.6 |
| 1100 | $cm'^2 b'^2(1-m'b')$ | 54.3 | 52.7 | $c^2 c'm'^2 b'$ | 41.9 | 39.2 | 69.9 |
| 0111 | $c'b^3$ | 39.4 | 38.1 | $c^2 c'(1-m'^2 b')$ | 71.0 | 74.4 | 67.6 |
| 1011 | $cm'b'(1-m'b')^2$ | 64.2 | 65.6 | $c^2 c'(1-m'^2 b')$ | 75.7 | 74.4 | 67.2 |
| 1101 | $cm'b'(1-m'b')^2$ | 65.3 | 65.6 | $c^2 c'(1-m'^2 b')$ | 74.7 | 74.4 | 70.2 |
| 1110 | $cm'b'(1-m'b')^2$ | 62.0 | 65.6 | $c^3 m'^3 b'$ | 33.8 | 35.8 | 72.2 |
| 1111 | $c(1-m'b')^3$ | 90.8 | 89.6 | $c^3(1-m'^3 b')$ | 91.0 | 90.0 | 75.6 |

Note. $c'=1-c$. $m'=1-m$. $b'=1-b$.

Causal networks not only predict pairwise correlations between directly connected features. Figure 2 indicates that as a result of the asymmetries inherent in causal relationships there is an important disanalogy between the common-cause and common-effect schemas: Although the common-cause schema implies that the three effects ($F_2$, $F_3$, $F_4$) will be correlated (albeit more weakly than directly connected features), the common-effect schema does not imply that the three causes ($F_1$, $F_2$, $F_3$) will be correlated. This asymmetry between common-cause and common-effect schemas has been the focus of considerable investigation in the philosophical and psychological literatures [3, 5]. Use of these schemas in the following experiment enables a test of whether categorizers are sensitive the pattern of correlations between features directly-connected by causal laws, and also those that arise due to the asymmetries inherent in causal relationships shown in Figure 2. Moreover, I will show that CMT predicts, and humans exhibit, sensitivity to interactions among features of a higher-order than the pairwise interactions shown in Figure 2.

## Method

Six novel categories were used in which the description of causal relationships between features consisted of one sentence indicating the cause and effect feature, and then one or two sentences describing the mechanism responsible for the causal relationship. For example, one of the novel categories, Lake Victoria Shrimp, was described as having four binary features (e.g., A high quantity of ACh neurotransmitter. , Long-lasting flight response. , Accelerated sleep cycle. , etc.)

and causal relationships among those features (e.g., "A high quantity of ACh neurotransmitter causes a long-lasting flight response. The duration of the electrical signal to the muscles is longer because of the excess amount of neurotransmitter.").

Participants first studied several computer screens of information about their assigned category at their own pace. All participants were first presented with the category's four features. Participants in the common-cause condition were additionally instructed on the common-cause causal relationships ($F_1 \rightarrow F_2$, $F_1 \rightarrow F_3$, $F_1 \rightarrow F_4$), and participants in the common-effect condition were instructed on the common-effect relationships ($F_1 \rightarrow F_4$, $F_2 \rightarrow F_4$, $F_3 \rightarrow F_4$). When ready, participants took a multiple-choice test that tested them on the knowledge they had just studied. Participants were required to retake the test until they committed 0 errors.

Participants then performed a classification task in which they rated on a 0-100 scale the category membership of 16 exemplars, consisting of all possible objects that can be formed from four binary features. For example, those participants assigned to learn the Lake Victoria Shrimp category were asked to classify a shrimp that possessed "High amounts of the ACh neurotransmitter," "A normal flight response," "Accelerated sleep cycle," and "Normal body weight." The order of the test exemplars was randomized for each participant.

One hundred and eight University of Illinois undergraduates received course credit for participating in this experiment. They were randomly assigned in equal numbers to the three conditions, and to one of the six experimental categories.

## Results

Categorization ratings for the 16 test exemplars averaged over participants in the common-cause, common-effect, and control conditions are presented in Table 1. The presence of causal knowledge had a large effect on the ratings. For instance, exemplars 0111 and 0001 were given lower ratings in the common-cause and common-effect conditions, respectively (39.4 and 26.3) than in the control condition (67.6 and 67.0) presumably because in these exemplars correlations are broken (effect features are present even though their causes are absent). In contrast, exemplar 1111 received a significantly higher rating in the common-cause and common-effect conditions than in the control condition (90.8 and 91.0 vs. 75.6), presumably because in both conditions all correlations are preserved.

To confirm that causal schemas induced a sensitivity to interactions between features, categorization ratings were analyzed by performing a multiple regression for each participant. Four predictor variables ($f_1$, $f_2$, $f_3$, $f_4$) were coded as -1 if the feature was absent, and +1 if it was present. An additional six predictor variables were formed from the multiplicative interaction between pairs of features: $f_{12}$, $f_{13}$, $f_{14}$, $f_{24}$, $f_{34}$, and $f_{23}$. For those feature pairs connected by a causal relationship the two-way interaction terms represent whether the causal relationship is confirmed (+1, cause and effect both present or both absent) or violated (-1, one present and one absent). Finally, the four three-way interactions ($f_{123}$, $f_{124}$, $f_{134}$, and $f_{234}$), and the single four-way interaction ($f_{1234}$) were also included as predictors.

Regression weights averaged over participants are presented in Figure 3 as a function of causal schema condition. Figure 3 indicates that the interaction terms corresponding to those feature pairs assigned causal relationships had significantly positive weights in both the common-cause condition ($f_{12}$, $f_{13}$, $f_{14}$), and the common-effect condition ($f_{14}$, $f_{24}$, $f_{34}$). That is, as predicted (Figure 2) an exemplar was rated a better category member when it preserved expected correlations (cause and effect feature either both present or both absent), and a worse member when it broke those correlations (one absent and the other present).

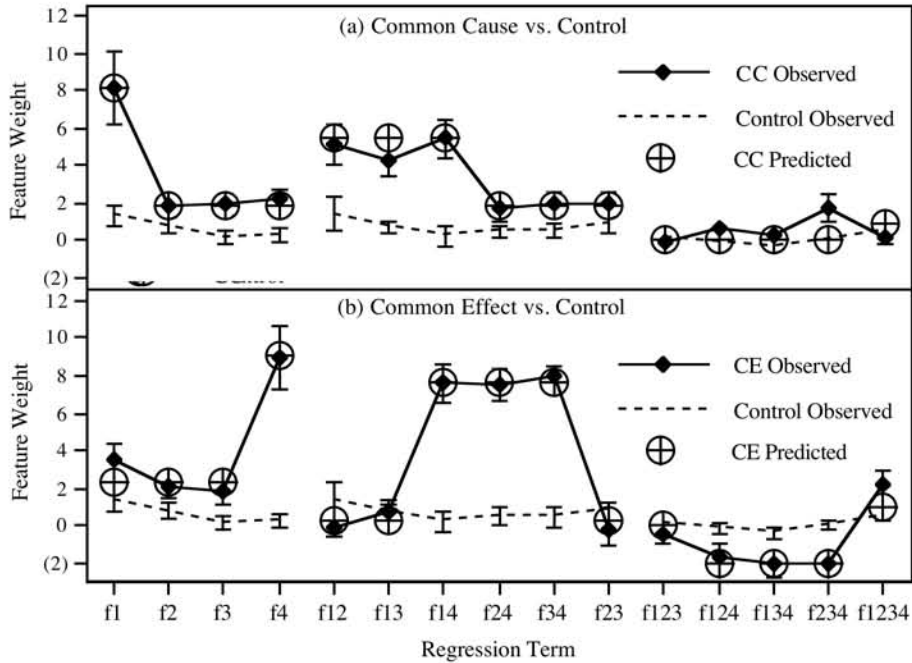

Figure 3

In addition, it was shown earlier (Figure 2) that because of their common-cause the three effect features in a common-cause schema will be correlated, albeit more weakly than directly-linked features. Consistent with this prediction, in this condition the three two-way interaction terms between the effect features ($f_{24}$, $f_{34}$, $f_{23}$) are greater than those interactions in the control condition. In contrast, the common-effect schema does not imply that the three cause features will be correlated, and in fact in that condition the interactions between the cause attributes ($f_{12}$, $f_{13}$, $f_{23}$) did not differ from those in the control condition (Figure 3).

Figure 3 also reveals higher-order interactions among features in the common-effect condition: Weights on interaction terms $f_{124}$, $f_{134}$, $f_{234}$, and $f_{1234}$ ($-1.6$, $2.0$, $-2.0$, and $2.2$) were significantly different from those in the control condition. These higher-order interactions arose because a common-effect schema requires only one cause feature to explain the presence of the common effect. Figures 7b presents the logarithm of the ratings in the common-effect condition for those test exemplars in which the common effect is present as a function of the number of cause features

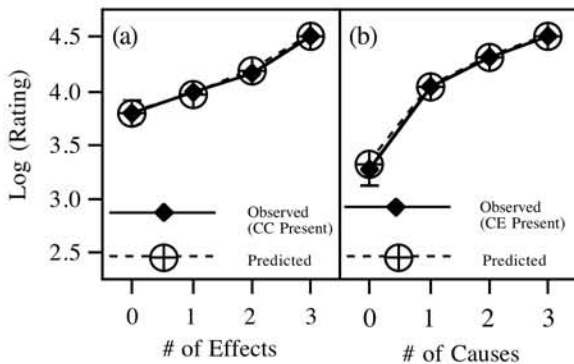

Figure 4

present. Ratings increased more with the introduction of the first cause as compared to subsequent causes. That is, participants considered the presence of at least one cause explaining the presence of the common-effect to be sufficient grounds to grant an exemplar a relatively high category membership rating in a common-effect category. In contrast, Figure 7a shows a linear

increase in (the logarithm of) categorization ratings for those exemplars in which the common cause is present as a function of the number of effect features. In the presence of the common cause each additional effect produced a constant increment to log categorization ratings.

Finally, Figure 3 also indicates that the simple feature weights differed as a function of causal schema. In the common-cause condition, the common-cause ($f_1$) carried greater weight than the three effects ($f_2$, $f_3$, $f_4$). In contrast, in the common-effect condition it was the common-effect ($f_4$) that had greater weight than the three causes ($f_1$, $f_2$, $f_3$). That is, causal networks promote the importance of not only specific feature combinations, but the importance of individual features as well.

## Model Fitting

To assess whether CMT accounts for the patterns of classification found in this experiment, the causal models of Figure 1 were fitted to the category membership ratings of each participant in the common-cause and common-effect conditions, respectively. That is, the ratings were predicted from the equation,

$$Rating\ (X) = K\ \yen\ Likelihood\ (X;\ c,\ m,\ b)$$

where *Likelihood* ($X$; $c$, $m$, $b$) is the likelihood of exemplar $X$ as a function of $c$, $m$, and $b$. The likelihood equations for the common-cause and common-effect models shown in Table 1 were used for common-cause and common-effect participants, respectively. $K$ is a scaling constant that brings the likelihood into the range 0-100. For each participant, the values for parameters $K$, $c$, $m$, and $b$ that minimized the squared deviation between the predicted and observed ratings was computed. The best fitting values for parameters $K$, $c$, $m$, and $b$ averaged over participants were 846, .578, .214, and .437 in the common-cause condition, and 876, .522, .325, and .280 in the common-effect condition. The predicted ratings for each exemplar are presented in Table 1. The significantly positive estimate for $m$ in both conditions indicates that participants categorization performance was consistent with them assuming the presence of a probabilistic causal mechanisms linking category features. Ratings predicted by CMT did not differ from observed ratings according to chi-square tests: $\chi^2(16)=3.0$ for common cause, $\chi^2(16)=5.3$ for common-effect.

To demonstrate that CMT predicts participants sensitivity to particular combinations of features when categorizing, each participants predicted ratings were subjected to the same regressions that were performed on the observed ratings. The resulting regression weights averaged over participants are presented in Figure 3 superimposed on the weights from the observed data. First, Figure 3 indicates that CMT reproduces participants sensitivity to agreement between pairs of features directly connected by causal relationships ($f_{12}$, $f_{13}$, $f_{14}$ in the common-cause condition, and $f_{14}$, $f_{24}$, $f_{34}$ in the common-effect condition). That is, according to both CMT and human participants, category membership ratings increase when pairs of features confirm causal laws, and decrease when they violate those laws. Second, Figure 3 indicates that CMT accounts for the interactions between the effect features in the common-cause condition ($f_{12}$, $f_{13}$, $f_{23}$) and also for the higher-order feature interactions in the common-effect condition ($f_{124}$, $f_{134}$, $f_{234}$, $f_{1234}$), indicating that that CMT is also sensitive to the asymmetries inherent in causal relationships. The predictions of CMT superimposed on the observed data in Figure 4 confirm that CMT, like the human participants, requires only one cause feature to  explain  the presence of a common effect (nonlinear increase in ratings in Figure 4b) whereas CMT predicts a linear increase in log ratings as one adds effect features to a common cause (Figure 4a). Finally, CMT also accounts for the larger weight given to the common cause and common-effect features  (Figure 3).

# Discussion

The current results support CMTs claims that people have a representation of the probabilistic causal mechanisms that link category features, and that they classify by evaluating whether an objects combination of features was likely to have been generated by those mechanisms. That is, people have models of the world that lead them to expect a certain distribution of features in category members, and consider exemplars good category members to the extent they manifest those expectations.

One way this effect manifested itself is in terms of the importance of preserved correlations between features directly connected by causal relationships. An alternative model that accounts for this particular result assumes that the feature space is expanded to include *configural cues* encoding the confirmation or violation of each causal relationship [6]. However, such a model treats causal links as symmetric and does not consider interactions among links. As a result, it does not fit the common effect data as well as CMT (Figure 4b), because it is unable to account for categorizers sensitivity to the higher-order feature interactions that emerge as a result of causal asymmetries in a complex network.

CMT diverges from traditional models of categorization by emphasizing the knowledge people possess as opposed to the examples they observe. Indeed, the current experiment differed from many categorization studies in not providing examples of category members. As a result, CMT is applicable to the many real-world categories about which people know far more than they have observed first hand (e.g., scientific concepts). Of course, for many other categories people observe category members, and the nature of the interactions between knowledge and observations is an open question of considerable interest. Using the same materials as in the current study, the effects of knowledge and observations have been orthogonally manipulated with the finding that observations had little effect on classification performance as compared to the theories [7]. Thus, theories may often dominate categorization decisions even when observations are available.

## Acknowledgments

Support for this research was provided by funds from the National Science Foundation (Grants Number SBR-9816458 and SBR 97-20304) and from the National Institute of Mental Health (Grant Number R01 MH58362).

## References

[1] Murphy, G. L., & Medin, D. L. (1985). The role of theories in conceptual coherence. *Psychological Review*, *92*, 289-316.

[2] Rehder, B. (1999). A causal model theory of categorization. In *Proceedings of the 21st Annual Meeting of the Cognitive Science Society* (pp. 595-600). Vancouver.

[3] Waldmann, M.R., Holyoak, K.J., & Fratianne, A. (1995). Causal models and the acquisition of category structure. *Journal of Experimental Psychology: General*, *124*, 181-206.

[4] Pearl, J. (1988). *Probabilistic reasoning in intelligent systems: Networks of plausible inference*. San Mateo, CA: Morgan Kaufman.

[5] Salmon, W. C. (1984). *Scientific explanation and the causal structure of the world*. Princeton, NJ: Princeton University Press.

[6] Gluck, M. A., & Bower, G. H. (1988). Evaluating an adaptive network model of human learning. *Journal of Memory and Language, 27*, 166-195.

[7] Rehder, B., & Hastie, R. (2001). Causal knowledge and categories: The effects of causal beliefs on categorization, induction, and similarity. *Journal of Experimental Psychology: General, 130*, 323-360.
